# An Estimation-Theoretic Framework for the Presentation of Multiple Stimuli

**Christian W. Eurich**[*]
Institute for Theoretical Neurophysics
University of Bremen
Otto-Hahn-Allee 1
D-28359 Bremen, Germany
`eurich@physik.uni-bremen.de`

## Abstract

A framework is introduced for assessing the encoding accuracy and the discriminational ability of a population of neurons upon simultaneous presentation of multiple stimuli. Minimal square estimation errors are obtained from a Fisher information analysis in an abstract compound space comprising the features of all stimuli. Even for the simplest case of linear superposition of responses and Gaussian tuning, the symmetries in the compound space are very different from those in the case of a single stimulus. The analysis allows for a quantitative description of attentional effects and can be extended to include neural nonlinearities such as nonclassical receptive fields.

## 1 Introduction

An important issue in the Neurosciences is the investigation of the encoding properties of neural populations from their electrophysiological properties such as tuning curves, background noise, and correlations in the firing. Many theoretical studies have used estimation theory, in particular the measure of Fisher information, to account for the neural encoding accuracy with respect to the presentation of a single stimulus (e. g., [1, 2, 3, 4, 5]).

Most modeling studies, however, neglect the fact that in a natural situation, neural activity results from multiple objects or even complex sensory scenes. In particular, attention experiments require the presentation of at least one distractor along with the attended stimulus. Electrophysiological data are now available demonstrating effects of selective attention on neural firing behavior in various cortical areas [6, 7, 8]. Such experiments require the development of theoretical tools which deviate from the usual practice of considering only single stimuli in the analysis. Zemel et al. [9] employ an extended encoding scheme for stimulus distributions and use Bayesian decoding to account for the presentation of multiple objects. Similarly, Bayesian estimation has been used in the context of attentional phenomena [10].

---

[*]homepage: http://www-neuro.physik.uni-bremen.de/~eurich

In this paper, a new estimation-theoretic framework for the simultaneous presentation of multiple stimuli is introduced. Fisher information is employed to compute lower bounds for the encoding error and the discriminational ability of neural populations independent of a particular estimator. Here we focus on the simultaneous presentation of two objects in the context of attentional phenomena. Furthermore, we assume a linearity in the neural response for reasons of analytical tractability; however, the method can be extended to include neural nonlinearities.

## 2 Estimation Theory for Multiple Stimuli

### 2.1 Tuning Curves in Compound Space

The tuning curve $f(\mathcal{X})$ of a neuron is defined to be the average neural response to repetitive presentations of stimulus configurations $\mathcal{X}$. In most cases, the response is taken to be the number $n(\mathcal{X})$ of action potentials occurring within some time interval $\tau$ after stimulus presentation, or the neural *firing rate* $r(\mathcal{X}) = n(\mathcal{X})/\tau$:

$$f(\mathcal{X}) = \langle r(\mathcal{X}) \rangle = \frac{\langle n(\mathcal{X}) \rangle}{\tau} . \tag{1}$$

Within an estimation-theoretic framework, the variability of the neural response is described by a probability distribution conditioned on the value of $\mathcal{X}$, $P(n; \mathcal{X})$. The average $\langle \cdot \rangle$ in (1) can be regarded either as an average over multiple presentations of the same stimulus configuration (in an experimental setup), or as an average over $n$ (in a theoretical description).

In most electrophysiological experiments, tuning curves are assessed through the presentation of a *single* stimulus, $\mathcal{X} = \vec{x}$, such as a bar or a grating characterized by a single orientation, or a dot of light at a specific position in the animal's visual field (e.g., [11, 12]). Such tuning curves will be denoted by $f_1(\vec{x})$, where the subscript refers to the single object.

The behavior of a neuron upon presentation of *multiple* objects, however, cannot be inferred from tuning curves $f_1(\vec{x})$. Instead, neurons may show nonlinearities such as the so-called non-classical receptive fields in the visual area V1 which have attracted much attention in the recent past (e. g., [13, 14]). For $M$ simultaneously presented stimuli, $\mathcal{X} = \vec{x}_1, \ldots, \vec{x}_M$, the neuronal tuning curve can be written as a function $f_M(\vec{x}_1, \ldots, \vec{x}_M)$, where the subscript $M$ is not necessarily a parameter of the function but an indicator of the number of stimuli it refers to. The domain of this function will be called the *compound space* of the stimuli.

In the following, we consider a specific example consisting of two simultaneously presented stimuli, characterized by a single physical property (such as orientation or direction of movement). The resulting tuning function is therefore a function of two scalar variables $x_1$ and $x_2$: $f_2(x_1, x_2) = \langle r(x_1, x_2) \rangle = \langle n(x_1, x_2) \rangle / \tau$. Figure 1 visualizes the concept of the compound space.

In order to obtain an analytical access to the encoding properties of a neural population, we will furthermore assume that a neuron's response $f_2(x_1, x_2)$ is a *linear superposition* of the single-stimulus responses $f_1(x_1)$ and $f_1(x_2)$, i. e.,

$$f_2(x_1, x_2) = k f_1(x_1) + (1 - k) f_1(x_2) , \tag{2}$$

where $0 < k < 1$ is a factor which scales the relative importance of the two stimuli. Such linear behavior has been observed in area 17 of the cat upon presentation of bi-vectorial transparent motion stimuli [15] and in areas MT and MST of the macaque monkey upon simultaneous presentation of two moving objects [16]. In

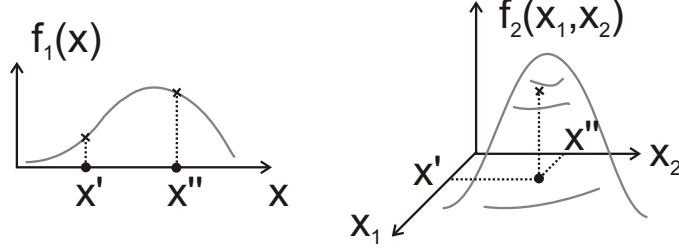

Figure 1: *The concept of compound space. A single-stimulus tuning curve $f_1(x)$ (left) yields the average response to the presentation of either $x'$ or $x''$; the simultaneous presentation of $x'$ and $x''$, however, can be formalized only through a tuning curve $f_2(x_1, x_2)$ (right).*

general, however, the compound space method is not restricted to linear neural responses.

The consideration of a neural population in the compound space yields tuning properties and symmetries which are very different from those in a $D$-dimensional single-stimulus space considered in the literature (e. g., [2, 3, 4]). First, the tuning curves have a different appearance. Figure 2a shows a tuning curve $f_2(x_1, x_2)$ given by (2), where $f_1(x)$ is a Gaussian,

$$f_1(x) = F \exp \left\{ -\frac{(x - c)^2}{2\sigma^2} \right\} ; \qquad (3)$$

$F$ is a gain factor which can be scaled to be the maximal firing rate of the neuron. $f_2(x_1, x_2)$ is not radially symmetric but has cross-shaped level curves. Second, a

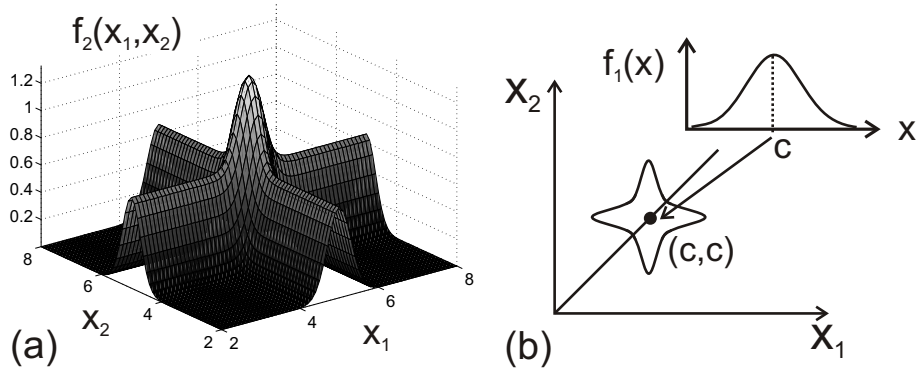

Figure 2: *(a) A tuning curve $f_2(x_1, x_2)$ in a 2-dimensional compound space given by (2) and (3) with $k = 0.5$, $c = 5$, $\sigma = 0.3$, $F = 1$. (b) Arrangement of tuning curves: The centers of the tuning curves are restricted to the diagonal $x_1 = x_2$. The cross is a schematic cross-section of the tuning curve in (a).*

single-stimulus tuning curve $f_1(x)$ whose center is located at $x = c$ yields a linear superposition whose center is given by the vector $(c, c)$ in the compound space. This is due to the fact that both axes describe the same physical stimulus feature. Therefore, all tuning curve centers are restricted to the 1-dimensional subspace

$x_1 = x_2$. The tuning curve centers are assumed to have a distribution in the compound space which can be written as

$$\tilde{\eta}(c_1, c_2) = \begin{cases} 0 & \text{if } c_1 \neq c_2 \\ \eta(c) & \text{if } c_1 = c_2 \end{cases}. \tag{4}$$

The geometrical features in the compound space suggest that an estimation-theoretic approach will yield encoding properties of neural populations which are different from those obtained from the presentation of a single stimulus.

## 2.2  Fisher Information

In order to assess the encoding accuracy of a neural population, the stochasticity of the neural response is taken into account. For $N$ neurons, it is formalized as the probability of obtaining $n^{(i)}$ spikes in the $i$-th neuron $(i = 1 \ldots, N)$ as a response to the stimulus configuration $\mathcal{X}$, $P(n^{(1)}, n^{(2)}, \ldots, n^{(N)}; \mathcal{X}) \equiv P(\vec{n}; \mathcal{X})$. Here we assume independent spike generation mechanisms in the neurons:

$$P(n^{(1)}, n^{(2)}, \ldots, n^{(N)}; \mathcal{X}) = \prod_{i=1}^{N} P(n^{(i)}; \mathcal{X}). \tag{5}$$

These parameter-dependent distributions are obtained either experimentally or through a noise model; a convenient choice for the latter is a Poisson distribution with a spike count average given by the tuning curve (1) of each neuron.

In the 2-dimensional compound space discussed in the previous section, $P(\vec{n}; \mathcal{X}) \equiv P(\vec{n}; x_1, x_2)$. The *Fisher information* is a $2 \times 2$ matrix $J(x_1, x_2) = (J_{ij}(x_1, x_2))$ $(i, j \in \{1, 2\})$, whose entries are given by

$$J_{ij}(x_1, x_2) = \left\langle \left( \frac{\partial}{\partial x_i} \ln P(\vec{n}; x_1, x_2) \right) \left( \frac{\partial}{\partial x_j} \ln P(\vec{n}; x_1, x_2) \right) \right\rangle \quad (i, j \in \{1, 2\}). \tag{6}$$

The Cramér-Rao inequality states that a lower bound on the expected square estimation error of the $i$th feature, $\epsilon_{i,\min}^2$ (i=1,2), is given by $(J^{-1})_{ii}$ provided that the estimator is unbiased. In the following, this lower bound is studied in the 2-dimensional compound space.

## 3  Results

**Single-neuron Fisher Information.**  The single-neuron Fisher information in the compound space can be written down for an arbitrary noise model. Here we choose a Poissonian spike distribution,

$$P(n; x_1, x_2) = \frac{(\tau f_2(x_1, x_2))^n \exp\{-\tau f_2(x_1, x_2)\}}{n!}, \tag{7}$$

whereby the tuning is assumed to be linear according to (2), and the single-stimulus tuning curve $f_1(x)$ is a Gaussian given by (3). A straightforward calculation yields the single-neuron Fisher information matrix $J^c(x_1, x_2) = (J_{ij}^c(x_1, x_2))$ $(i, j \in \{1, 2\})$ given by

$$J^c(x_1, x_2) = \frac{\tau F}{\sigma^4 \left\{ ke^{-\frac{(x_1-c)^2}{2\sigma^2}} + (1-k)e^{-\frac{(x_2-c)^2}{2\sigma^2}} \right\}} \times \tag{8}$$

$$\begin{pmatrix} k^2(x_1-c)^2 e^{-\frac{(x_1-c)^2}{\sigma^2}} & k(1-k)(x_1-c)(x_2-c)e^{-\frac{(x_1-c)^2+(x_2-c)^2}{2\sigma^2}} \\ k(1-k)(x_1-c)(x_2-c)e^{-\frac{(x_1-c)^2+(x_2-c)^2}{2\sigma^2}} & (1-k)^2(x_2-c)^2 e^{-\frac{(x_2-c)^2}{\sigma^2}} \end{pmatrix};$$

the index $c$ refers to the center $(c, c)$ of the tuning curve.

**Population Fisher Information.** For independently spiking neurons (5), the population Fisher information is the sum of the single-neuron Fisher information values. Assuming some density $\eta(c)$ of tuning curve centers on the diagonal $x_1 = x_2$, the population Fisher information is therefore obtained by an integration of (8). Here we consider the simple case of a constant density, $\eta(c) \equiv \eta_0$ resulting in elements $J_{ij}(x_1, x_2)$ $(i, j \in \{1, 2\})$ of the Fisher information maxtrix given by

$$J_{ij}(x_1, x_2) = \eta \int_{-\infty}^{\infty} J_{ij}^c(x_1, x_2) dc. \tag{9}$$

A symmetry with respect to the diagonal $x_1 = x_2$ allows the replacement of the two variable $x_1$, $x_2$ by a single variable $\rho$ visualized in Fig. 3. It is straightforward

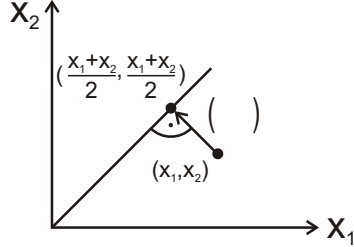

Figure 3: *Transformation to the variable $\rho$ which is proportional to the distance of the point $(x_1, x_2)$ to the diagonal. $\rho$ therefore quantifies the similarity of the stimuli $x_1$ and $x_2$.*

to obtain two additional symmetries, $J_{12}(\rho) = J_{21}(\rho)$ and $J_{11}(\rho) = J_{11}(-\rho)$. The final population Fisher information is given by

$$J(\rho) = \begin{pmatrix} J_{11}(\rho) & J_{12}(\rho) \\ J_{12}(\rho) & \frac{(1-k)^2}{k^2} J_{11}(\rho) \end{pmatrix}, \tag{10}$$

whereby

$$J_{11}(\rho) = \frac{k^2 \tau F \eta}{\sigma} \int_{-\infty}^{\infty} \frac{(\xi + \frac{\rho}{\sigma})^2 \exp\{-(\xi + \frac{\rho}{\sigma})^2\}}{k \exp\{-\frac{1}{2}(\xi + \frac{\rho}{\sigma})^2\} + (1-k) \exp\{-\frac{1}{2}(\xi - \frac{\rho}{\sigma})^2\}} d\xi,$$

$$J_{12}(\rho) = \frac{k(1-k) \tau F \eta}{\sigma} \int_{-\infty}^{\infty} \frac{(\xi + \frac{\rho}{\sigma})(\xi - \frac{\rho}{\sigma}) \exp\{-\frac{1}{2}((\xi + \frac{\rho}{\sigma})^2 + (\xi - \frac{\rho}{\sigma})^2)\}}{k \exp\{-\frac{1}{2}(\xi + \frac{\rho}{\sigma})^2\} + (1-k) \exp\{-\frac{1}{2}(\xi - \frac{\rho}{\sigma})^2\}} d\xi.$$

In the following, three examples will be discussed.

### 3.1 Example 1: Symmetrical Tuning

First we study the symmetrical case $k = 1/2$ the receptive fields of which are given in Fig. 2a. Fig. 4 shows the minimal square estimation error for $x_1$, $\epsilon_{1,\min}^2(\rho)$, as obtained from the first diagonal element of the inverse Fisher information matrix. Due to the symmetry, it is identical to the minimal square error for $x_2$, $\epsilon_{2,\min}^2(\rho)$. The estimation error diverges as $\rho \longrightarrow 0$. This can be understood as follows: For $k = 1/2$, the matrix (10) is symmetric and can be diagonalized. The eigenvector directions are

$$\vec{v}_1 = \frac{1}{\sqrt{2}} \begin{pmatrix} 1 \\ 1 \end{pmatrix} \qquad \vec{v}_2 = \frac{1}{\sqrt{2}} \begin{pmatrix} -1 \\ 1 \end{pmatrix}. \tag{11}$$

Correspondingly, the diagonal Fisher information matrix yields a lower bound for the estimation errors of $(x_1 + x_2)/\sqrt{2}$ and $(x_2 - x_1)/\sqrt{2}$, respectively. The results are shown in Fig. 5. The estimation error for $(x_1 + x_2)/\sqrt{2}$ takes a finite value for

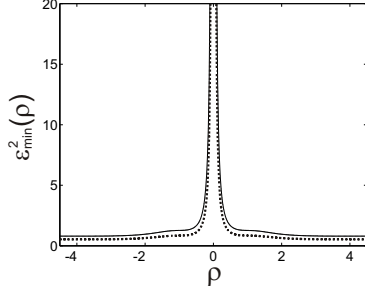

Figure 4: *Minimal square estimation error for stimulus $x_1$ or $x_2$. Solid line: $F = 1$; dotted line: $F = 1.5$. In both cases, $k = 0.5$, $\sigma = 1$, $\tau = 1$, $\eta = 1$.*

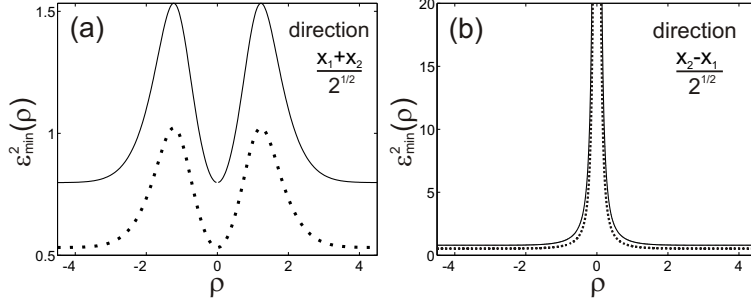

Figure 5: *Minimal square estimation error for (a) $(x_1 + x_2)/\sqrt{2}$ and (b) $(x_2 - x_1)/\sqrt{2}$. Solid lines: $F = 1$; dotted lines: $F = 1.5$. Same parameters as in Fig. 4.*

all $\varrho$. However, the estimation error for $(x_2 - x_1)/\sqrt{2}$ diverges as $\rho \longrightarrow 0$. This error corresponds to an estimation of the *difference* of the two presented stimuli. As expected, a discrimination becomes impossible as the stimuli merge. The Fisher information for $(x_2 - x_1)/\sqrt{2}$ can be regarded as a discrimination measure which takes the simultaneous presentation of stimuli into account.

## 3.2   Example 2: Attention on Both Stimuli

Electrophysiological studies in V1 and V4 [7] and MT [8] of macaque monkeys suggest that the gain but not the width of tuning curves is increased as stimuli in a cell's receptive field are attended. This can easily be incorporated in the current model: The gain corresponds to the factor $F$ in the tuning curve (3). Figures 4 and 5 compare the results obtained in the previous section ($F = 1$) with a maximal firing rate $F = 1.5$. As expected, the minimal square errors are smaller for higher $F$ in all cases (dotted lines); a higher firing rate yields a better stimulus estimation. This suggests that attention increases localization accuracy of $x_1$ and $x_2$ as well as their discrimination if both stimuli are attended. The former is consistent with psychophysical results on attentional enhancement of spatial resolution in human subjects [17].

## 3.3   Example 3: Attending One Stimulus

The situation changes if only one of the two stimuli is attended. Electrophysiological recordings in monkey area V4 suggest that upon presentation of two stimuli inside a neuron's receptive field, the influence of the attended stimulus increases as compared to the unattended one [6]. In our framework, this situation can be considered by increasing the weight factor of the attended stimulus in the linear

superposition (2). Here we study the case $k = 0.75$ corresponding to attending stimulus $x_1$. The resulting tuning curve shows characteristic distortions as compared to the symmetrical case $k = 0.5$ (Fig. 6a). The Fisher information analysis

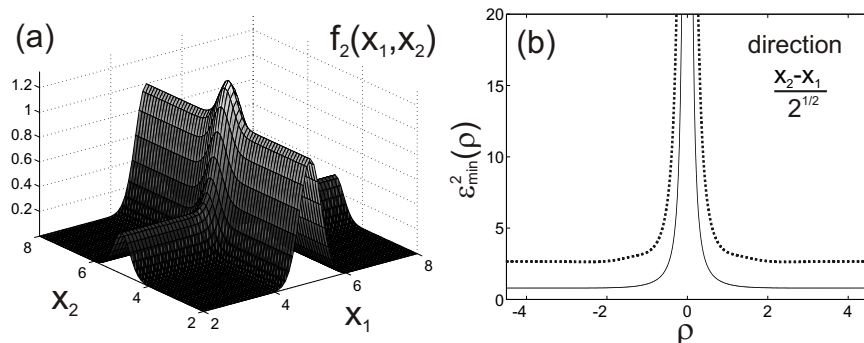

Figure 6: *Neural encoding for one attended stimulus. (a) Tuning curve (2), (3) for $k = 0.75$, i.e., stimulus $x_1$ is attended. All other parameters as in Fig. 1a. (b) Minimal square estimation errors for the direction $(x_2 - x_1)/\sqrt{2}$ resulting from a rotated Fisher information matrix. Solid line: $k = 0.5$ as in Fig. 5b; dotted line: $k = 0.75$. $F = 1$, all other parameters as in Fig. 4.*

reveals that the attended stimulus $x_1$ yields a smaller minimal square estimation error than it does in the non-attention case $k = 0.5$ whereas the minimal square error for the unattended stimulus $x_2$ is increased (data not shown). Figure 6b shows the minimal square error for the difference of the stimuli, $(x_2 - x_1)/\sqrt{2}$. The minimal estimation error becomes *larger* as compared to $k = 0.5$. This result can be interpreted as follows: Attending stimulus $x_1$ yields a better encoding of $x_1$ but a worse encoding of $x_2$. The latter results in the larger estimation error for the difference $(x_2 - x_1)/\sqrt{2}$ of the stimulus values. This can be interpreted as a worse discriminational ability: In a psychophysical experiment, subjects attending stimulus $x_1$ will have only a crude representation of the unattended stimulus $x_2$ will therefore yield a performance which is worse as compared to the situation where both stimuli are processed in the same way. This is a prediction resulting from the presented framework.

## 4   Summary and Discussion

A method was introduced to account for the encoding of multiple stimuli by populations of neurons. Estimation theory was performed in a compound space whose axes are defined by the features of each stimulus. Here we studied a specific example of linear neurons with Gaussian tuning and Poissonian spike statistics to gain insight into the symmetries in the compound space and the interpretation of the resulting estimation errors. The approach allows for a detailed consideration of attention effects on the neural level [7, 8, 6]. The method can be extended to include nonlinear neural behavior as multiple stimuli are presented; see e.g. [13, 14], where the response of single neurons to two orientation stimuli cannot be easily inferred from the neural behavior in the case of only one stimulus. More experimental and theoretical work has to be done in order to account for the psychophysical performance under the influence of attention as it has been measured, for example, in [17]. For this purpose, the presented approach has to be related to classical measures in

discrimination and same-different tasks. From theoretical considerations in the case of a single stimulus [2, 3, 4, 5] it is well known that the encoding accuracy of a neural population may depend on various properties such as the number of encoded features, the noise model, and the correlations in the neural activity. The influence of such factors within the presented framework is currently under investigation.

**Acknowledgments**

I wish to thank Shun-ichi Amari, Hiroyuki Nakahara, Anthony Marley and Stefan Wilke for stimulating discussions. Part of this paper was written during my stay at the RIKEN institute. I also acknowledge support from SFB 517, Neurocognition.

# References

[1] M. A. Paradiso, *A theory for the use of visual orientation information which exploits the columnar structure of striate cortex*, Biol. Cybern. **58** (1988) 35–49.

[2] K. Zhang and T. J. Sejnowski, *Neuronal tuning: to sharpen or broaden?* Neural Comp. **11** (1999) 75–84.

[3] C. W. Eurich and S. D. Wilke, *Multidimensional encoding strategy of spiking neurons*, Neural Comp. **12** (2000) 1519–1529.

[4] S. D. Wilke and C. W. Eurich, *Representational accuracy of stochastic neural populations*, Neural Comp. **14** (2001) 155–189.

[5] H. Nakahara, S. Wu and S.-i. Amari, *Attention modulation of neural tuning through peak and base rate*, Neural Comp. **13** (2001) 2031–2047.

[6] J. Moran and R. Desimone, *Selective attention gates visual processing in the extrastriate cortex*, Science **229** (1985) 782–784.

[7] C. J. McAdams and J. H. R. Maunsell, *Effects of attention on orientation-tuning functions of single neurons in macaque cortical area V4*, J. Neurosci. **19** (1999) 431–441.

[8] S. Treue and J. C. Martínetz Trujillo, *Feature-based attention influences motion processing gain in macaque visual cortex*, Nature **399** (1999) 575–579.

[9] R. S. Zemel, P. Dayan and A. Pouget, *Probabilistic interpretation of population codes*, Neural Comp. **10** (1998) 403–430.

[10] P. Dayan and R. S. Zemel, *Statistical models and sensory attention*, in: D. Willshaw und A. Murray (eds), *Procedings of the Ninth International Conference on Artificial Neural Networks, ICANN 99*, Venue, University of Edinburgh (1999) 1017–1022.

[11] D. H. Hubel and T. Wiesel, *Receptive fields and functional architecture of monkey striate cortex*, J. Physiol. **195** (1968) 215–244.

[12] N. V. Swindale (1998), *Orientation tuning curves: empirical description and estimation of parameters*, Biol. Cybern. **78** (1998) 45–56.

[13] J. J. Knierim und D. van Essen, *Neuronal responses to static texture patterns in area V1 of the alert macaque monkey*, J. Neurophysiol. **67** (1992) 961–979.

[14] A. M. Sillito, K. Grieve, H. Jones, J. Cudeiro und J. Davies, *Visual cortical mechanisms detecting focal orientation discontinuities*, Nature **378** (1995) 492–496.

[15] R. J. A. van Wezel, M. J. M. Lankheet, F. A. J. Verstraten, A. F. M. Marée and W. A. van de Grind, *Responses of complex cells in area 17 of the cat to bi-vectorial transparent motion*, Vis. Res. **36** (1996) 2805–2813.

[16] G. H. Recanzone, R. H. Wurtz and U. Schwarz, *Responses of MT and MST neurons to one and two moving objects in the receptive field*, J. Neurophysiol. **78** (1997) 2904–2915.

[17] Y. Yeshurun and M. Carrasco, *Attention improves or impairs visual performance by enhancing spatial resolution*, Nature **396** (1998) 72–75.
